# Learning curves for Gaussian processes regression: A framework for good approximations

**Dörthe Malzahn**          **Manfred Opper**
Neural Computing Research Group
School of Engineering and Applied Science
Aston University, Birmingham B4 7ET, United Kingdom.
[malzahnd,opperm]@aston.ac.uk

## Abstract

Based on a statistical mechanics approach, we develop a method for approximately computing average case learning curves for Gaussian process regression models. The approximation works well in the large sample size limit and for arbitrary dimensionality of the input space. We explain how the approximation can be systematically improved and argue that similar techniques can be applied to general likelihood models.

## 1  Introduction

Gaussian process (GP) models have gained considerable interest in the Neural Computation Community (see e.g.[1, 2, 3, 4] ) in recent years. Being non-parametric models by construction their theoretical understanding seems to be less well developed compared to simpler parametric models like neural networks. We are especially interested in developing theoretical approaches which will at least give good approximations to generalization errors when the number of training data is sufficiently large.

In this paper we present a step in this direction which is based on a statistical mechanics approach. In contrast to most previous applications of statistical mechanics to learning theory we are not limited to the so called "thermodynamic" limit which would require a high dimensional input space.

Our work is very much motivated by recent papers of Peter Sollich (see e.g. [5]) who presented a nice approximate treatment of the Bayesian generalization error of GP regression which actually gives good results even in the case of a one dimensional input space. His method is based on an exact recursion for the generalization error of the regression problem together with approximations that decouple certain correlations of random variables. Unfortunately, the method seems to be limited because the exact recursion is an artifact of the Gaussianity of the regression model and is not available for other cases such as classification models. Second, it is not clear how to assess the quality of the approximations made and how one may systematically improve on them. Finally, the calculation is (so far) restricted to

a full Bayesian scenario, where a prior average over the unknown data generating function simplifies the analysis.

Our approach has the advantage that it is more general and may also be applied to other likelihoods. It allows us to compute other quantities besides the generalization error. Finally, it is possible to compute the corrections to our approximations.

## 2   Regression with Gaussian processes

To explain the Gaussian process scenario for regression problems [2], we assume that we observe corrupted values $y(x) \in R$ of an unknown function $f(x)$ at input points $x \in R^d$. If the corruption is due to independent Gaussian noise with variance $\sigma^2$, the likelihood for a set of $m$ example data $D = (y(x_1), \ldots, y(x_m)))$ is given by

$$P(D|f) = \frac{\exp\left(-\sum_{i=1}^m \frac{(y_i - f(x_i))^2}{2\sigma^2}\right)}{(2\pi\sigma^2)^{\frac{m}{2}}}. \tag{1}$$

where $y_i \doteq y(x_i)$. The goal of a learner is to give an estimate of the function $f(x)$. The available prior information is that $f$ is a realization of a Gaussian process (random field) with zero mean and covariance $C(x, x') = E[f(x)f(x')]$, where $E$ denotes the expectation over the Gaussian process. We assume that the prediction at a test point $x$ is given by the posterior expectation of $f(x)$, i.e. by

$$\hat{f}(x) = E\{f(x)|D\} = \frac{Ef(x)P(D|f)}{Z} \tag{2}$$

where the partition function $Z$ normalises the posterior. Calling the true data generating function $f^*$ (in order to distinguish it from the functions over which we integrate in the expectations) we are interested in the learning curve, i.e. the generalization (mean square) error averaged over independent draws of example data, i.e. $\varepsilon_g = [\langle (f^*(x) - \hat{f}(x))^2 \rangle]_D$ as a function of $m$, the sample size. The brackets $[\ldots]_D$ denote averages over *example* data sets where we assume that the inputs $x_i$ are drawn independently at random from a density $p(x)$. $\langle \ldots \rangle$ denotes an average over *test* inputs drawn from the same density. Later, the same brackets will also be used for averages over several different test points and for joint averages over test inputs and test outputs.

## 3   The Partition Function

As typical of statistical mechanics approaches, we base our analysis on the averaged "free energy" $[-\ln Z]_D$ where the partition function $Z$ (see Eq. (2)) is

$$Z = EP(D|f). \tag{3}$$

$[\ln Z]_D$ serves as a generating function for suitable posterior averages. The concrete application to $\varepsilon_g$ will be given in the next section. The computation of $[\ln Z]_D$ is based on the replica trick $\ln Z = \lim_{n \to 0} \frac{Z^n - 1}{n}$, where we compute $[Z^n]_D$ for integer $n$ and perform the continuation at the end.

Introducing a set of auxiliary integration variables $z_{ka}$ in order to decouple the squares, we get

$$[Z^n]_D = \int \prod_{k=1}^m \prod_{a=1}^n \frac{dz_{ka}}{2\pi} e^{-\frac{\sigma^2}{2}\sum_{k,a} z_{ka}^2} E_n \left[ \exp\left(i \sum_{k,a} z_{ka}(f_a(x_k) - y_k)\right) \right]_D \tag{4}$$

where $E_n$ denotes the expectation over the $n$ times replicated GP measure. In general, it seems impossible to perform the average over the data. Using a cumulant expansion, an infinite series of terms would be created. However one may be tempted to try the following heuristic approximation: If (for fixed function $f$), the distribution of $f(x_k) - y_k$ was a zero mean Gaussian, we would simply end up with only the second cumulant and

$$[Z^n]_D \approx \int \prod_{k,a} \frac{dz_{ka}}{2\pi} \exp\left(-\frac{\sigma^2}{2}\sum_{k,a} z_{ka}^2\right) \times \tag{5}$$

$$\times \quad E_n \exp\left(-\frac{1}{2}\sum_{a,b}\sum_k z_{ka}z_{kb}\langle(f_a(x) - y)(f_b(x) - y)\rangle\right).$$

Although such a reasoning may be justified in cases where the dimensionality of inputs $x$ is large, the assumption of approximate Gaussianity is typically (in the sense of the prior measure over functions $f$) completely wrong for small dimensions. Nevertheless, we will argue in the next section that the expression Eq. (5) (justified by a different reason) is a good approximation for large sample sizes and nonzero noise level. We will postpone the argument and proceed to evaluate Eq. (5) following a fairly standard recipe: The high dimensional integrals over $z_{ka}$ are turned into low dimensional integrals by the introduction of "order-parameters" $\eta_{ab} = \sum_{k=1}^m z_{ka}z_{kb}$ so that

$$[Z^n]_D \approx \int \prod_{a\leq b} d\eta_{ab} \exp\left(-\frac{1}{2}\sigma^2 \sum_a \eta_{aa} + G(\{\eta\})\right) \times \tag{6}$$

$$\times \quad E_n \exp\left(-\frac{1}{2}\sum_{a,b} \eta_{ab}\langle(f_a(x) - y)(f_b(x) - y)\rangle\right)$$

where $e^{G(\{\eta\})} = \int \prod_{k,a} \frac{dz_{ka}}{2\pi} \prod_{a\leq b}\delta\left(\sum_{k=1}^m z_{ka}z_{kb} - \eta_{ab}\right)$. We expect that in the limit of large sample size $m$, the integrals are well approximated by the saddle-point method. To perform the limit $n \to 0$, we make the assumption that the saddle-point of the matrix $\eta$ is replica symmetric, i.e. $\eta_{ab} = \eta$ for $a \neq b$ and $\eta_{aa} = \eta_0$. After some calculations we arrive at

$$[\ln Z]_D = -\frac{\sigma^2 \eta_0}{2} + \frac{m}{2}\ln(\eta_0 - \eta) + \frac{m\eta}{2(\eta_0 - \eta)} - \frac{\eta}{2}\langle E^0 f^2(x)\rangle \tag{7}$$

$$+ \quad \ln E \exp\left[-\frac{\eta_0 - \eta}{2}\langle(f(x) - y)^2\rangle\right] - \frac{m}{2}\left(\ln(2\pi m) - 1\right)$$

into which we have to insert the values $\eta$ and $\eta_0$ that make the right hand side an extremum. We have defined a new auxiliary (translated) Gaussian measure over functions by

$$E^0\{\phi\{f\}\} = \frac{E \exp\left[-\frac{\eta_0 - \eta}{2}\langle f^2(x)\rangle\right]\phi\{f\}}{E \exp\left[-\frac{\eta_0 - \eta}{2}\langle f^2(x)\rangle\right]} \tag{8}$$

where $\phi$ is a functional of $f$. For a given input distribution it is possible to compute the required expectations in terms of sums over eigenvalues and eigenfunctions of the covariance kernel $C(x, x')$. We will give the details as well as the explicit order parameter equations in a full version of the paper.

## 4 Generalization error

To relate the generalization error with the order parameters, note that in the replica framework (assuming the approximation Eq. (5)) we have

$$
\varepsilon_g + \sigma^2 = -\lim_{n \to 0} \int \prod_{a \leq b} d\eta_{ab} \, \exp\left[-\frac{1}{2}\sigma^2 \sum_a \eta_{aa} + G(\{\eta\})\right] \times
$$

$$
\times \frac{\partial}{\partial \eta_{12}} E_n \exp\left(-\frac{1}{2}\sum_{a,b} \eta_{ab}\langle (f_a(x) - y)(f_b(x) - y)\rangle\right)
$$

which by a partial integration and a subsequent saddle point integration yields

$$
\varepsilon_g = -\frac{m\eta}{(\eta_0 - \eta)^2} - \sigma^2. \tag{9}
$$

It is also possible to compute other error measures in terms of the order parameters like the expected error on the (noisy) training data defined by

$$
\varepsilon_t = \frac{1}{m}\sum_i [(y_i - \hat{f}(x_i))^2]_D = -\frac{\sigma^4 \eta}{m}. \tag{10}
$$

The "true" training error which compares the prediction with the data generating function $f^*$ is somewhat more complicated and will be given elsewhere.

## 5 Why (and when) the approximation works

Our intuition behind the approximation Eq. (5) is that for sufficiently large sample size, the partition function is dominated by regions in function space which are close to the data generating function $f^*$ such that terms like $\langle (f_a(x) - y)(f_b(x) - y)\rangle$ are typically small and higher order polynomials in $f_a(x) - y$ generated by a cumulant expansion are less important. This intuition can be checked self consistently by estimating the omitted terms perturbatively. We use the following modified partition function

$$
[Z^n(\lambda)]_D = \int \prod_{k,a} \frac{dz_{ka}}{2\pi} \, e^{-\frac{\sigma^2}{2}\sum_{k,a} z_{ka}^2} E_n\left[\exp\left(i\lambda \sum_{k,a} z_{ka}(f_a(x_k) - y)\right.\right.
$$

$$
\left.\left. - \frac{1-\lambda^2}{2}\sum_{a,b}\sum_k z_{ka}z_{kb}\langle (f_a(x) - y)(f_b(x) - y)\rangle\right)\right]_D \tag{11}
$$

which for $\lambda = 1$ becomes the "true" partition function, whereas Eq. (5) is obtained for $\lambda = 0$. Expanding in powers of $\lambda$ (the terms with odd powers vanish) is equivalent to generating the cumulant expansion and subsequently expanding the non-quadratic terms down. Within the saddle-point approximation, the first nonzero correction to our approximation of $[\ln Z]$ is given by

$$
\lambda^4 \left(\frac{(\eta_0 - \eta)^2}{2m}\left(\sigma^2\langle \hat{C}(x,x)\rangle + \langle \hat{C}(x,x)F^2(x)\rangle - \langle \hat{C}(x,x')F(x)F(x')\rangle\right.\right.
$$

$$
\left. + \eta\langle \hat{C}(x,x')\hat{C}(x,x'')\hat{C}(x',x'')\rangle - \eta\langle \hat{C}(x,x)\hat{C}^2(x,x')\rangle\right)
$$

$$
\left. + \frac{1}{4}\left(-\frac{\eta^2}{m} + \frac{\eta_0^2}{m}\right)\left(\langle \hat{C}^2(x,x)\rangle - \langle \hat{C}^2(x,x')\rangle\right)\right). \tag{12}
$$

$\hat{C}(x,x') = E^0\{f(x)f(x')\}$ denotes the covariance with respect to the auxiliary measure and $F(x) \doteq f^*(x) - \langle\hat{C}(x,x'')f^*(x'')\rangle$. The significance of the individual terms as $m \to \infty$ can be estimated from the following scaling. We find that $(\eta_0 - \eta) = \mathcal{O}(m)$ is a positive quantity, whereas $\eta = \mathcal{O}(m)$ is negative. $\hat{C}(x,x') = \mathcal{O}(1/m)$. Using these relations, we can show that Eq. (12) remains finite as $m \to \infty$, whereas the leading approximation Eq. (7) diverges with $m$.

We have not (yet) computed the resulting correction to $\varepsilon_g$. However, we have studied the somewhat simpler error measure $\varepsilon' \doteq \frac{1}{m}\sum_i[E\{(f^*(x_i) - f(x_i))^2|D\}]_D$ which can be obtained from a derivative of $[\ln Z]_D$ with respect to $\sigma^2$. It equals the error of a Gibbs algorithm (sampling from the posterior) on the training data. We can show that the correction to $\varepsilon'$ is typically by a *factor* of $\mathcal{O}(1/m)$ *smaller* than the leading term. However, our approximation becomes worse with decreasing noise variance $\sigma^2$. $\sigma = 0$ is a singular case for which (at least for some GPs with slowly decreasing eigenvalues) it can be shown that our approximation for $\varepsilon_g$ decays to zero at the wrong rate. For small values of $\sigma$, $\sigma \to 0$, we expect that higher order terms in the perturbation expansion will become relevant.

## 6 Results

We compare our analytical results for the error measures $\varepsilon_g$ and $\varepsilon_t$ with simulations of GP regression. For simplicity, we have chosen periodic processes of the form $f(x) = \sqrt{2}\sum_n (a_n\cos(2\pi nx) + b_n\sin(2\pi nx))$ for $x \in [0,1]$ where the coefficients $a_n, b_n$ are independent Gaussians with $E\{a_n^2\} = E\{b_n^2\} = \lambda_n$. This choice is convenient for analytical calculations by the orthogonality of the trigonometric functions when we sample the $x_i$ from a uniform density in $[0,1]$. The $\lambda_n$ and the translation invariant covariance kernel are related by $c(x - y) \doteq C(x,y) = 2\sum_n \lambda_n\cos(2\pi n(x - y))$ and $\lambda_n = \int_0^1 c(x)\cos(2\pi nx)\,dx$. We specialise on the (periodic) RBF kernel $c(x) = \sum_{k=-\infty}^\infty \exp\left[-(x - k)^2/2l^2\right]$ with $l = 0.1$. For an illustration we generated learning curves for two target functions $f^*$ as displayed in Fig. 1. One function is a sine-wave $f^*(x) = \sqrt{2\lambda_1}\sin(2\pi x)$ while the other is a random realisation from the prior distribution. The symbols in the left panel of Fig. 1 represent example sets of fifty data points. The data points have been obtained by corruption of the target function with Gaussian noise of variance $\sigma^2 = 0.01$. The right panel of Fig. 1 shows the data averaged generalization and training errors $\varepsilon_g$, $\varepsilon_t$ as a function of the number $m$ of example data. Solid curves display simulation results while the results of our theory Eqs. (9), (10) are given by dashed lines. The training error $\varepsilon_t$ converges to the noise level $\sigma^2$. As one can see from the pictures our theory is very accurate when the number $m$ of example data is sufficiently large. While the generalization error $\varepsilon_g$ differs initially, the asymptotic decay is the same.

## 7 The Bayes error

We can also apply our method to the Bayesian generalization error (previously approximated by Peter Sollich [5]). The Bayes error is obtained by averaging the generalization error over "true" functions $f^*$ drawn at random from the prior distribution. Within our approach this can be achieved by an average of Eq. (7) over $f^*$. The resulting order parameter equations and their relation to the Bayes error turn out be *identical* to Sollich's result. Hence, we have managed to re-derive his approximation within a broader framework from which also possible corrections can be obtained.

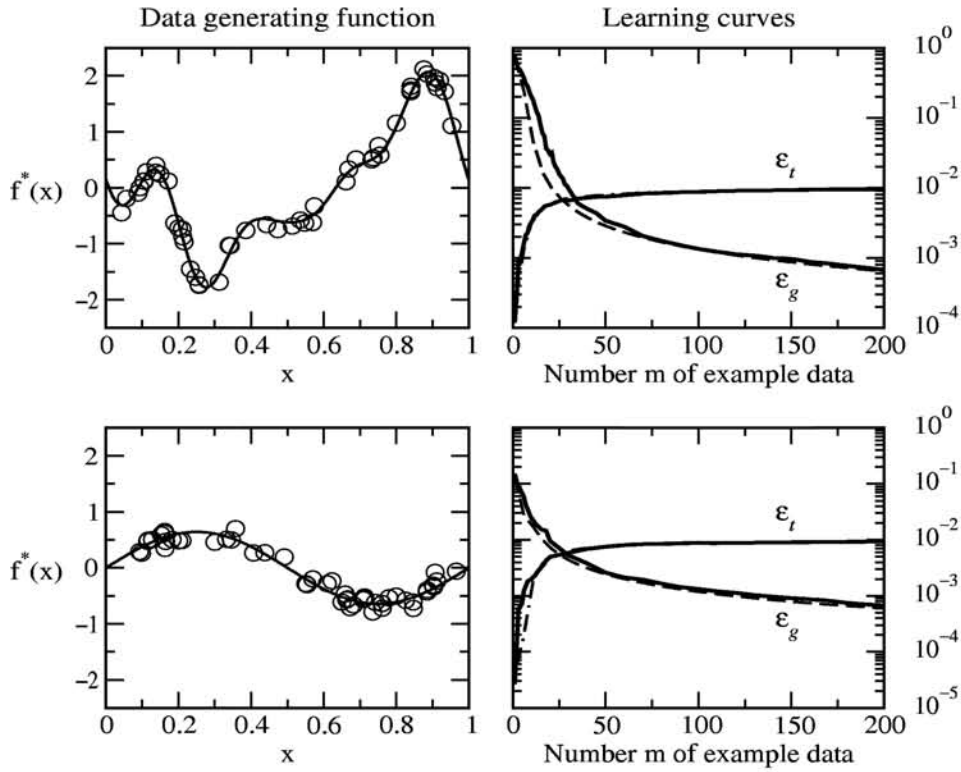

Figure 1: The left panels show two data generating functions $f^*(x)$ and example sets of 50 data points. The right panels display the corresponding averaged learning curves. Solid curves display simulation results for generalization and training errors $\varepsilon_g$, $\varepsilon_t$. The results of our theory Eqs. (9), (10) are given by dashed lines.

## 8  Future work

At present, we extend our method in the following directions:

- The statistical mechanics framework presented in this paper is based on a partition function $Z$ which can be used to generate a variety of other data averages for posterior expectations. An obvious interesting quantity is given by the sample fluctuations of the generalization error

$$\left[\langle\langle(f^*(x)-\hat{f}(x))^2\rangle^2\rangle\right]_D - \left([\langle\langle(f^*(x)-\hat{f}(x))^2\rangle\rangle]_D\right)^2 \qquad (13)$$

which gives confidence intervals on $\varepsilon_g$.

- Obviously, our method is not restricted to a regression model (in this case however, all resulting integrals are elementary) but can also be directly generalized to other likelihoods such as the classification case [4, 6]. A further application to Support Vector Machines should be possible.

- The saddle-point approximation neglects fluctuations of the order parameters. This may be well justified when $m$ is sufficiently large. It is possible to improve on this by including the quadratic expansion around the saddle-point.

- Finally, one may criticise our method as being of minor relevance to practical applications, because our calculations require the knowledge of the unknown function $f^*$ and the density of the inputs $x$. However, Eqs. (9) and (10) show that important error measures are solely expressed by the order parameters $\eta$ and $\eta_0$. Hence, estimating some error measures and the posterior variance at the data points empirically would allow us to predict values for the order parameters. Those in turn could be used to make predictions for the unknown generalization error.

## Acknowledgement

This work has been supported by EPSRC grant GR/M81601.

## References

[1] D. J. C. Mackay, Gaussian Processes, A Replacement for Neural Networks, NIPS tutorial 1997, May be obtained from http://wol.ra.phy.cam.ac.uk/pub/mackay/.

[2] C. K. I. Williams and C. E. Rasmussen, Gaussian Processes for Regression, in *Neural Information Processing Systems 8*, D. S. Touretzky, M. C. Mozer and M. E. Hasselmo eds., 514-520, MIT Press (1996).

[3] C. K. I. Williams, Computing with Infinite Networks, in *Neural Information Processing Systems 9*, M. C. Mozer, M. I. Jordan and T. Petsche, eds., 295-301. MIT Press (1997).

[4] D. Barber and C. K. I. Williams, Gaussian Processes for Bayesian Classification via Hybrid Monte Carlo, in *Neural Information Processing Systems 9*, M . C. Mozer, M. I. Jordan and T. Petsche, eds., 340-346. MIT Press (1997).

[5] P. Sollich, Learning curves for Gaussian processes, in *Neural Information Processing Systems 11*, M. S. Kearns, S. A. Solla and D. A. Cohn, eds. 344 - 350, MIT Press (1999).

[6] L. Csató, E. Fokoué, M. Opper, B. Schottky, and O. Winther. Efficient approaches to Gaussian process classification. In *Advances in Neural Information Processing Systems*, volume 12, 2000.
